# On-Chip Compensation of Device-Mismatch Effects in Analog VLSI Neural Networks

**Miguel Figueroa**
Department of Electrical Engineering, Universidad de Concepción
Casilla 160-C, Correo 3, Concepción, Chile
*mfigueroa@die.udec.cl*

**Seth Bridges and Chris Diorio**
Computer Science & Engineering, University of Washington
Box 352350, Seattle, WA 98195-2350, USA
{*seth, diorio*}*@cs.washington.edu*

## Abstract

Device mismatch in VLSI degrades the accuracy of analog arithmetic circuits and lowers the learning performance of large-scale neural networks implemented in this technology. We show compact, low-power on-chip calibration techniques that compensate for device mismatch. Our techniques enable large-scale analog VLSI neural networks with learning performance on the order of 10 bits. We demonstrate our techniques on a 64-synapse linear perceptron learning with the Least-Mean-Squares (LMS) algorithm, and fabricated in a $0.35\mu$m CMOS process.

## 1 Introduction

Modern embedded and portable electronic systems operate in unknown and mutating environments, and use adaptive filtering and machine learning techniques to discover the statistics of the data and continuously optimize their performance. Artificial neural networks are an attractive substrate for implementing these techniques, because their regular computation and communication structures makes them a good match for custom VLSI implementations. Portable systems operate under severe power dissipation and space constraints, and VLSI implementations provide a good tradeoff between computational throughput and power/area cost. More specifically, analog VLSI neural networks perform their computation using the physical properties of transistors with orders of magnitude less power and die area than their digital counterparts. Therefore, they could enable large-scale real-time adaptive signal processing systems on a single die with minimal power dissipation.

Despite the promises delivered by analog VLSI, an important factor has prevented the success of large-scale neural networks using this technology: device mismatch. Gradients in the parameters of the fabrication process create variations in the physical properties of silicon devices across a single chip. These variations translate into gain and offset mismatches in the arithmetic blocks, which severely limit the overall performance of the system. As a result, the accuracy of analog implementations rarely exceeds 5-6 bits, even for small-scale networks. This limitation renders these implementations useless for many important applications. Although it is possible to combat some of these effects using careful design techniques, they come at the cost of increased power and area, making an analog solution less attractive.

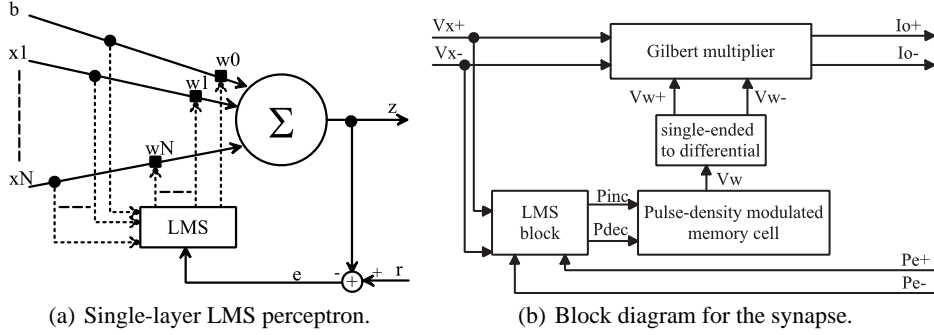

(a) Single-layer LMS perceptron.   (b) Block diagram for the synapse.

Figure 1: A single-layer perceptron and synapse. (a) The output $z$ of the perceptron is the inner product between the input and weight vectors. The LMS algorithm updates the weights based on the inputs and an error signal $e$. (b) The synapse stores the weight in an analog memory cell. A Gilbert multiplier computes the product between the input and the weight and outputs a differential current. The LMS block updates the weight.

We have built a 64-synapse analog neural network with an learning performance of 10 bits, representing an improvement of more than one order of magnitude over that of traditional analog designs, with a modest increase in power and die area. We fabricated our network using a double-poly, 4-metal $0.35\mu$m CMOS process available from MOSIS. We achieve this performance by locally calibrating the critical analog blocks after circuit fabrication using a combination of one-time (or periodic) and continuous calibration using the same feedback as the network's learning algorithm. We chose the Least Mean Squares (LMS) algorithm because of its simplicity and wide applicability in supervised learning techniques such as adaptive filtering, adaptive inverse control, and noise canceling. Moreover, several useful unsupervised-learning techniques, such as adaptive orthogonalization, principal components analysis (PCA), independent components analysis (ICA) and decision-feedback learning, use simple generalizations of LMS.

## 2   A linear LMS perceptron

Fig. 1(a) shows our system architecture, a linear perceptron with scalar output that performs the function:

$$z(i) = bw_0(i) + \sum_{j=1}^{N} x_j(i)\, w_j(i) \tag{1}$$

where $i$ represents time, $z(i)$ is the output, $x_j(i)$ are the inputs, $w_j(i)$ are the synaptic weights, and $b$ is a constant bias input. We clarify the role of $b$ in Section 3.1. After each presentation of the input, the LMS algorithm updates the weights using the learning rule:

$$w_j(i+1) = w_j(i) + \eta\, x_j(i)\, e(i) \qquad i = 0\ldots N, x_0(i) = b \tag{2}$$

where $\eta$ is a constant learning rate, and $e(i)$ is the error between the output and a reference signal $r(i)$ such that $e(i) = r(i) - z(i)$.

## 3   The synapse

Fig. 1(b) shows a block diagram of our synapse. We store the synaptic weights in a memory cell that implements nonvolatile analog storage with linear updates. A circuit transforms the single-ended voltage output of the memory cell ($V_w$) into a differential voltage signal ($V_w^+, V_w^-$), with a constant common mode. A Gilbert multiplier computes the 4-quadrant product between this signal and the input (also represented as a differential voltage $V_x^+$, $V_x^-$). The output is a differential analog current pair ($I_o^+, I_o^-$), which we sum across all synapses by connecting them to common wires.

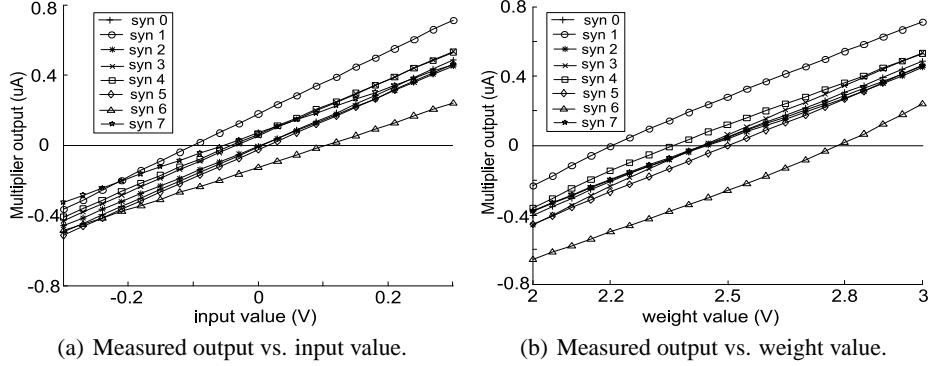

(a) Measured output vs. input value.　　　(b) Measured output vs. weight value.

Figure 2: Gilbert multiplier response for 8 synapses.(a) Our multiplier maximizes the linearity of $x_i$, achieving a linear range of 600mV differential. Gain mismatch is 2:1 and offset mismatch is up to 200mV. (b) Our multiplier maximizes weight range at the cost of weight linearity (1V single-ended, 2V differential). The gain variation is lower, but the offset mismatch exceeds 60% of the range.

Because we represent the perceptron's output and the reference with differential currents, we can easily compute the error using simple current addition. We then transform (off-chip in our current implementation) the resulting analog error signal using a pulse-density modulation (PDM) representation [1]. In this scheme, the value of the error is represented as the difference between the density (frequency) of two fixed-width, fixed-amplitude digital pulse trains ($P_e^+$ and $P_e^-$ in Fig. 1(b)). These properties make the PDM representation largely immune to amplitude and jitter noise. The performance of the perceptron is highly sensitive to the resolution of the error signal; therefore the PDM representation is a good match for it. The LMS block in the synapse takes the error and input values and computes update pulses (also using PDM) according to Eqn. 2.

In the rest of this section, we analyze the effects of device mismatch in the performance of the major blocks, discuss their impact in overall system performance, and present the techniques that we developed to deal with them. We illustrate with experimental results taken from silicon implementation of the perceptron in a $0.35\mu$m CMOS process. All data presented in this paper, unless otherwise stated, comes from this silicon implementation.

### 3.1   Multiplier

A Gilbert multiplier implements a nonlinear function of the product between two differential voltages. Device mismatch in the multiplier has two main effects: First, it creates offsets in the inputs. Second, mismatch across the entire perceptron creates variations in the offsets, gain, and linearity of the product. Thus, Eqn. 1 becomes:

$$z(i) = \sum_{j=0}^{N} a_j \left[ f_j^x \left( x_j(i) - d_j^x \right) \ f_j^w \left( w_j(i) - d_j^w \right) \right] \qquad x_0(i) = b \qquad (3)$$

where $a_j$ represents the gain mismatch between multipliers, $f_j^x$ and $f_j^w$ are the nonlinearities applied to the inputs and weights (also mismatched across the perceptron), and $d_j^x$ and $d_j^w$ are the mismatched offsets of the inputs and weights.

Our analysis and simulations of the LMS algorithm [2] determine that the performance of the algorithm is much more sensitive to the linearity of $f_j^x$ than to the linearity of $f_j^w$, because the inputs vary over their dynamic range with a large bandwidth, while the bandwidth of the weights is much lower than the adaptation time-constant. Therefore, the adaptation compensates for mild nonlinearities in the weights as long as $f_j^w$ remains a monotonic odd function [2]. Consequently, we sized the transistors in the Gilbert multiplier to maximize the linearity of $f_j^x$, but paid less attention (in order to minimize size and power) to $f_j^w$. Fig. 2(a) shows the output of 8 synapses in the system as a function of the input value. The

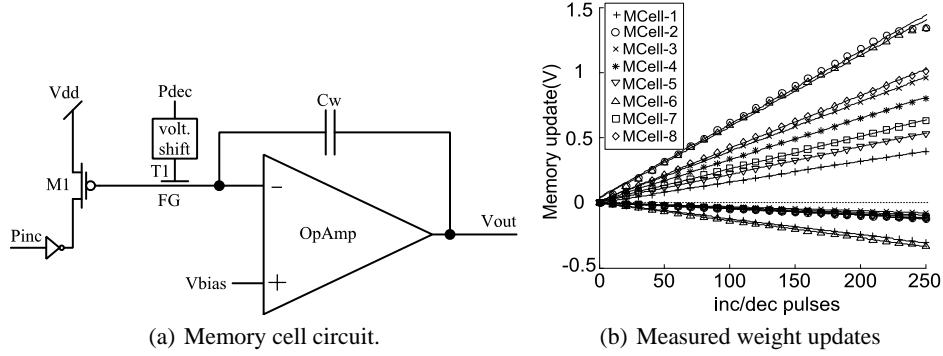

|                         |                          |
|-------------------------|--------------------------|
| (a) Memory cell circuit. | (b) Measured weight updates |

Figure 3: A simple PDM analog memory cell. (a) We store each weight as nonvolatile analog charge on the floating gate FG. The weight increments and decrements are proportional to the density of the pulses on $P_{inc}$ and $P_{dec}$. (b) Memory updates as a function of the increment and decrement pulse densities for 8 synapses. The updates show excellent linearity (10 bits), but also poor matching both within a synapse and between synapses.

response is highly linear. The gain mismatch is about 2:1, but the LMS algorithm naturally absorbs it into the learned weight value. Fig. 2(b) shows the multiplier output as a function of the single-ended weight value $V_w$. The linearity is visibly worse in this case, but the LMS algorithm compensates for it.

The graphs in the Fig. 2 also show the input and weight offsets. Because of the added mismatch in the single-ended to differential converter, the weights present an offset of up to $\pm 300$mV, or 30% of the weight range. The LMS algorithm will also compensate for this offset by absorbing it into the weight, as shown in the analysis of [3] for backprogagation neural networks. However, this will only occur if the weight range is large enough accomodate for the offset mismatch. Consequently, we sacrifice weight linearity to increase the weight range. Input offsets pose a harder problem, though. The offsets are small (up 100mV), but because of the restricted input range (to maximize linearity), they are large enough to dramatically affect the learning performance of the perceptron. Our solution was to use the bias synapse $w_0$ to compensate for the accumulated input offset. Assuming that the multiplier is linear, offsets translate into nonzero-mean inputs, which a bias synapse trained with LMS can remove as demonstrated in [4]. To guarantee sufficient gain, we provide a stronger bias current to the multiplier in the bias synapse.

## 3.2 Memory cell

A *synapse transistor* [5] is a silicon device that provides compact, accurate, nonvolatile analog storage as charge on its floating gate. Fowler-Nordheim tunneling adds charge to the floating gate and hot-electron injection removes charge. Both mechanisms can be used to accurately update the stored value during normal device operation. Because of these properties, synapse transistors have been a popular choice for weight storage in recent silicon learning systems [6, 7].

Despite the advantages listed above, it is hard to implement linear learning rules such as LMS using tunneling and injection. This is because their dynamics are exponential with respect to their control variables (floating-gate voltage, tunneling voltage and injection drain current), which naturally lead to weight-dependent nonlinear update rules. This is an important problem because the learning performance of the perceptron is strongly dependent on the accuracy of the weight updates; therefore distortions in the learning rule will degrade performance. The initial design of our memory cell, shown in Fig. 3(a) and based on the work presented in [8], solves this problem: We store the analog weight as charge on the floating gate FG of synapse transistor $M_1$. Pulses on $P_{dec}$ and $P_{inc}$ activate tunneling and injection and add or remove charge from the floating gate, respectively. The operational

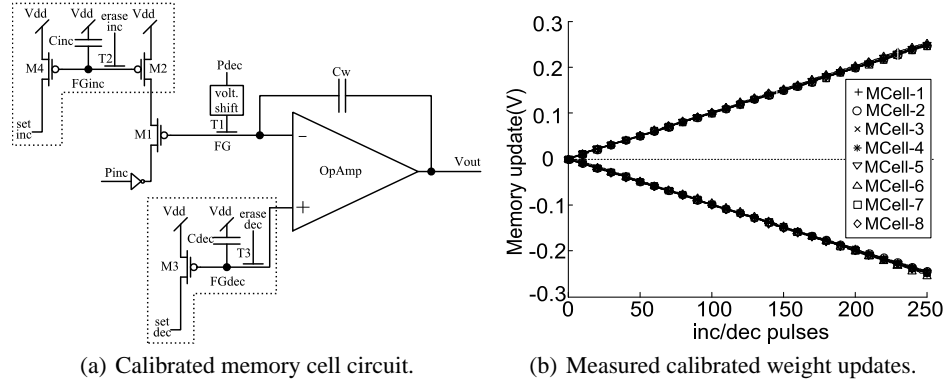

(a) Calibrated memory cell circuit.　　　(b) Measured calibrated weight updates.

Figure 4: PDM memory cell with local calibration. (a) We first match the tunneling rate across all synapses by locally changing the voltage at the floating gate $FG_{dec}$. Then, we modify the injection rate to match the local tunneling rate using the floating gate $FG_{inc}$. (b) The calibrated updates are symmetric and uniform within 9-10 bits.

amplifier sets the floating-gate voltage at the global voltage $V_{bias}$. Capacitor $C_w$ integrates the charge updates, changing the output $V_{out}$ by $\Delta V_{out} = \Delta Q/C$. Because the floating-gate voltage is constant and so are the pulse widths and amplitudes, the magnitude of the updates depends on the density of the pulses $P_{inc}$ and $P_{dec}$. Fig. 3(b) shows the magnitude of the weight updates as a function of the density of pulses in $P_{inc}$ (positive slopes) and $P_{dec}$ (negative slopes) for 8 synapses. The linearity of the updates, measured as the integral nonlinearity (INL) of the transfer functions depicted in Fig. 3(b), exceeds 10 bits.

Fig. 3(b) highlights an important problem caused by device mismatch: the strengths of tunneling and injection are poorly balanced within a synapse (the slopes show up to a 4:1 mismatch). Moreover, they show a variation of more than 3:1 across different synapses in the perceptron. This translates into asymmetric update rules that are also nonuniform across synapses. The local asymmetry of the learning rate translates into offsets between the learned and target weights, degrading the learning performance of the perceptron. The nonuniformity between learning rates across the perceptron changes Eqn. 2 into:

$$w_j(i+1) = w_j(i) + \eta_j\, x_j(i)\, e(i) \qquad i = 0 \ldots N, x_0(i) = b \tag{4}$$

where $\eta_j$ are the different learning rates for each synapse. Generalizing the conventional stability analysis of LMS [9], we can show that the condition for the stability of the weight vector is: $0 < \eta_{max} < 1/\lambda_{max}$, where $\lambda_{max}$ is the maximal eigenvalue of the input's correlation matrix and $\eta_{max} = max_j(\eta_j)$. Therefore, learning rate mismatch does not affect the accuracy of the learned weights, but it does slow down convergence because we need to scale all learning rates globally to limit the value of the maximal rate.

To maintain good learning performance and convergence speed, we need to make learning rates symmetric and uniform across the perceptron. We modified the design of the memory cell to incorporate local calibration mechanisms that achieve this goal. Fig. 4(a) shows our new design. The first step is to equalize tunneling rates: The voltage at the new floating gate $FG_{dec}$ sets the voltage at the floating-gate FG and controls the ratio between the strength of tunneling and injection onto FG: Raising the voltage at $FG_{dec}$ increases the drain-to-channel voltage and reduces the gate-to-tunneling-junction voltage at $M_1$, thus increasing injection efficiency and reducing tunneling strength [5]. We set the voltage at $FG_{dec}$ by first tunneling using the global line *erase_dec*, and then injecting on transistor $M_3$ by lowering the local line *set_dec* to equalize the tunneling rates across all synapses. To compare the tunneling rates, we issue a fixed number of pulses at $P_{dec}$ and compare the memory cell outputs using a double-sampling comparator (off-chip in the current implementation). To control the injection rate, we add transistor $M_2$, which limits the current through $M_1$ and

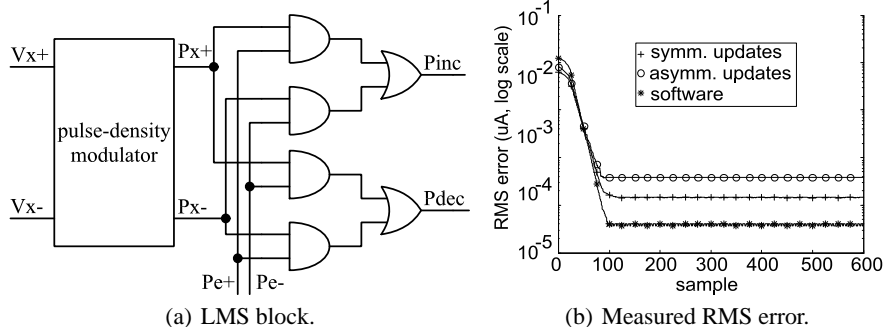

(a) LMS block.  (b) Measured RMS error.

Figure 5: LMS block at each synapse. (a) The difference between the densities of $P_{inc}$ and $P_{dec}$ is proportional to the product between the input and the error, and thus constitutes an LMS update rule. (b) RMS error for a single-synapse with a constant input and reference, including a calibrated memory cell with symmetric updates, a simple synapse with asymmetric updates, and a simulated ideal synapse.

thus the injection strength of the pulse at $P_{inc}$. We control the current limit with the voltage at the new floating gate $FG_{inc}$: we first remove electrons from the floating gate using the global line *erase_inc*. Then we inject on transistor $M_4$ by lowering the local line *set_inc* to match the injection rates across all synapses. The entire process is controlled by a simple state machine (also currently off-chip). Fig. 4(b) shows the tunneling and injection rates after calibration as a function of the density of pulses $P_{inc}$ and $P_{dec}$. Comparing the graph to Fig. 4(b), it is clear that the update rates are now symmetric and uniform across all synapses (they match within 9-10 bits). Note that we could also choose to calibrate just for learning rate symmetry and not uniformity across synapses, thus eliminating the floating gate $FG_{inc}$ and its associated circuitry. This optimization would result in approximately a 25% reduction in memory cell area (6% reduction in total synapse area), but would also cause an increase of more than 200% in convergence time, as illustrated in Section 4.

### 3.3 The LMS block

Fig. 5(a) shows a block diagram of the LMS-update circuit at each synapse. A pulse-density modulator [10] transforms the synaptic input into a pair of digital pulse-trains of fixed width ($P_x^+$, $P_x^-$). The value of the input is represented as the difference between the density (frequency) of the pulse trains. We implement the memory updates of Eqn. 2 by digitally combining the input and error pulses ($P_e^+$, $P_e^-$) such that:

$$P_{inc} = (P_x^+ \ AND \ P_e^+) \ OR \ (P_x^- \ AND \ P_e^-) \tag{5}$$
$$P_{dec} = (P_x^+ \ AND \ P_e^-) \ OR \ (P_x^- \ AND \ P_e^+) \tag{6}$$

This technique was used previously in a synapse-transistor based circuit that learns correlations between signals [11], and to multiply and add signals [1]. If the pulse trains are asynchronous and sparse, then using Eqn. 5 and Eqn. 6 to increment and decrement the synaptic weight implements the LMS learning rule of Eqn. 2.

To validate our design, we first trained a single synapse with a DC input to learn a constant reference. Because the input is constant, the linearity and offsets in the input signal do not affect the learning performance; therefore this experiment tests the resolution of the feedback path (LMS circuit and memory cell) isolated from the analog multipliers. Fig. 5(b) shows the evolution of the RMS value of the error for a synapse using the original and calibrated memory cells. The resolution of the pulse-density modulators is about 8 bits, which limits the resolution of the error signal. We also show the RMS error for a simulated (ideal) synapse learning from the same error. We plot the results in a logarithmic scale to highlight the differences between the three curves. The RMS error of the calibrated synapse converges to about 0.1nA. Computing the equivalent resolution in bits as

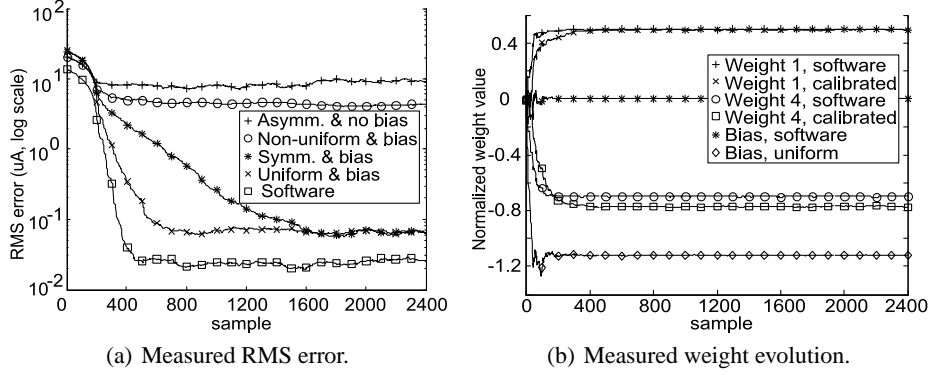

(a) Measured RMS error.          (b) Measured weight evolution.

Figure 6: Results for 64-synapse experiment. (a) Asymmetric learning rates and multiplier offsets limit the output resolution to around 3 bits. Symmetric learning rates and a bias synapse brings the resolution up to more 10 bits, and uniform updates reduce convergence time. (b) Synapse 4 shows a larger mismatch than synapse 1 and therefore it deviates from its theoretical target value to compensate. The bias synapse in the VLSI perceptron converges to a value that compensates for offsets in the inputs $x_i$ to the multipliers.

$r_b = -log_2 \left(0.5 \frac{RMS\ error}{output\ range}\right)$, we find that for a $2\mu$A output range, this error represents an output resolution of about 13 bits. The difference with the simulated synapse is due to the discrete weight updates in the PDM memory cell. Without calibration, the RMS error converges to 0.4nA (or about 11 bits), due to the offset in the learned weights introduced by the asymmetry in the learning rate. As discussed in Section 4, the degradation of the learning performance in a larger-scale system due to asymmetric learning rates is drastically larger.

## 4 A 64-synapse perceptron

To test our techniques in a larger-scale system, we fabricated a 64-synapse linear perceptron in a $0.35\mu$m CMOS process. The circuit uses $0.25$mm$^2$ of die area and dissipates $200\mu$W. Fig. 6(a) shows the RMS error of the output in a logarithmic scale as we introduce different compensation techniques. We used random zero-mean inputs selected from a uniform distribution over the entire input range, and trained the network using the response from a simulated perceptron with ideal multipliers and fixed weights as a reference. In our first experiments, we trained the network without using any compensation. The error settles to $10\mu$A RMS, which corresponds to an output resolution of about 3 bits for a full range of $128\mu$A differential. Calibrating the synapses for symmetric learning rates only improves the RMS error to $5\mu$A (4 bits), but the error introduced by the multiplier offsets still dominates the residual error. Introducing the bias synapse and keeping the learning rates symmetric (but nonuniform across the perceptron) compensates for the offsets and brings the error down to 60nA RMS, corresponding to an output resolution better than 10 bits. Further calibrating the synapses to achieve uniform, symmetric learning rates maintains the same learning performance, but reduces convergence time to less than one half, as predicted by the analysis in Section 3.2. A simulated software perceptron with ideal multipliers and LMS updates that uses an error signal of the same resolution as our experiments gives an upper bound of just under 12 bits for the learning performance.

Fig. 6(b) depicts the evolution of selected weights in the silicon perceptron with on-chip compensation and the software version. The graph shows that synapse 1 in our VLSI implementation suffers from little mismatch, and therefore its weight virtually converges to the theoretical value given by the software implementation. Because the PDM updates are discrete, the weight shows a larger oscillation around its target value than the software version. Synapse 4 shows a larger mismatch; therefore it converges to a visibly different value from the theoretical in order to compensate for it. The bias weight in the software percep-

tron converges to zero because the inputs have zero mean. In the VLSI perceptron, input offsets in the multipliers create nonzero-mean inputs; therefore the bias synapse converges to a value that compensates for the aggregated effect of the offsets. The normalized value of -1.2 reflects the gain boost given to this multiplier to increase its dynamic range.

## 5 Conclusions

Device mismatch prevents analog VLSI neural networks from delivering good learning performance for large-scale applications. We identified the key effects of mismatch and presented on-chip compensation techniques. Our techniques rely both on one-time (or periodic) calibration, and on the adaptive operation of the system to achieve continuous calibration. Combining these techniques with careful circuit design enables an improvement of more than one order of magnitude in accuracy compared to traditional analog designs, at the cost of an off-line calibration phase and a modest increase in die area and power. We illustrated our techniques with a 64-synapse analog-VLSI linear perceptron that adapts using the LMS algorithm. Future work includes extending these techniques to unsupervised learning algorithms such as adaptive orthogonalization, principal components analysis (PCA) and independent components analysis (ICA).

### Acknowledgements

This work was financed in part by the Chilean government through FONDECYT grant #1040617. We fabricated our chips through MOSIS.

## References

[1] Y. Hirai and K. Nishizawa, "Hardware implementation of a PCA learning network by an asynchronous PDM digital circuit," in *IEEE-INNS-ENNS International Joint Conference on Neural Networks (IJCNN)*, vol. 2, pp. 65–70, 2000.

[2] M. Figueroa, *Adaptive Signal Processing and Correlational Learning in Mixed-Signal VLSI*. Ph.D. Thesis, University of Washington, 2005.

[3] B. K. Dolenko and H. C. Card, "Tolerance to analog hardware of on-chip learning in back-propagation networks," *IEEE Transactions on Neural Networks*, vol. 6, no. 5, pp. 1045–1052, 1995.

[4] F. Palmieri, J. Zhu, and C. Chang, "Anti-Hebbian learning in topologically constrained linear networks: A tutorial," *IEEE Transactions on Neural Networks*, vol. 4, no. 5, pp. 748–761, 1993.

[5] C. Diorio, P. Hasler, B. Minch, and C. Mead, "A complementary pair of four-terminal silicon synapses," *Analog Integrated Circuits and Signal Processing*, vol. 13, no. 1/2, pp. 153–166, 1997.

[6] C. Diorio, D. Hsu, and M. Figueroa, "Adaptive CMOS: from biological inspiration to systems-on-a-chip," *Proceedings of the IEEE*, vol. 90, no. 3, pp. 345–357, 2002.

[7] J. Dugger and P. Hasler, "Improved correlation learning rule in continuously adapting floating-gate arrays using logarithmic pre-distortion of input and learning signals," in *IEEE Intl. Symposium on Circuits and Systems (ISCAS)*, vol. 2, pp. 536–539, 2002.

[8] C. Diorio, S. Mahajan, P. Hasler, B. A. Minch, and C. Mead, "A high-resolution nonvolatile analog memory cell," in *IEEE Intl. Symp. on Circuits and Systems*, vol. 3, pp. 2233–2236, 1995.

[9] B. Widrow and E. Walach, *Adaptive Inverse Control*. Upper Saddle River, NJ: Prentice-Hall, 1996.

[10] C. Mead, *Analog VLSI and Neural Systems*. Reading, MA: Addison-Wesley, 1989.

[11] A. Shon, D. Hsu, and C. Diorio, "Learning spike-based correlations and conditional probabilities in silicon," in *Neural Information Processing Systems (NIPS)*, (Vancouver, BC), 2001.
